# Repeat Until Bored: A Pattern Selection Strategy

**Paul W. Munro**
Department of Information Science
University of Pittsburgh
Pittsburgh, PA   15260

## ABSTRACT

An alternative to the typical technique of selecting training examples independently from a fixed distribution is formulated and analyzed, in which the current example is presented repeatedly until the error for that item is reduced to some criterion value, $\beta$; then, another item is randomly selected. The convergence time can be dramatically increased or decreased by this heuristic, depending on the task, and is very sensitive to the value of $\beta$.

## 1   INTRODUCTION

In order to implement the back propagation learning procedure (Werbos, 1974; Parker, 1985; Rumelhart, Hinton and Williams, 1986), several issues must be addressed. In addition to designing an appropriate network architecture and determining appropriate values for the learning parameters, the batch size and a scheme for selecting training examples must be chosen. The batch size is the number of patterns presented for which the corresponding weight changes are computed before they are actually implemented; immediate update is equivalent to a batch size of one. The principal pattern selection schemes are independent selections from a stationary distribution (independent identically distributed, or i.i.d.) and epochal, in which the training set is presented cyclically (here, each cycle through the training set is called an epoch). Under i.i.d. pattern selection, the learning performance is sensitive to the sequence of training examples. This observation suggests that there may exist selection strategies that facilitate learning. Several studies have shown the benefit of strategic pattern selection (e.g., Mozer and Bachrach, 1990; Atlas, Cohn, and Ladner, 1990; Baum and Lang, 1991).

Typically, online learning is implemented by independent identically distributed pattern selection, which cannot (by definition) take advantage of useful sequencing strategy. It seems likely, or certainly plausible, that the success of learning depends to some extent on the order in which stimuli are presented. An extreme, though negative, example would be to restrict learning to a portion of the available training set; i.e. to reduce the effective training set. Let sampling functions that depend on the state of the learner in a constructive way be termed *pedagogical*.

Determination of a particular input may require information exogenous to the learner; that is, just as training algorithms have been classified as supervised and unsupervised, so can pedagogical pattern selection techniques. For example, selection may depend on the network's performance relative to a desired schedule. The intent of this study is to explore an unsupervised selection procedure (even though a supervised learning rule, backpropagation, is used). The initial selection heuristic investigated was to evaluate the errors across the entire pattern set for each iteration and to present the pattern with the highest error; of course, this technique has a large computational overhead, but the question was whether it would reduce the number of learning trials. The results were quite to the contrary; preliminary trials on small tasks (two and three bit parity), show that this scheme performs very poorly with all patterns maintaining high error.

A new unsupervised selection technique is introduced here. The "Repeat-Until-Bored" heuristic is easily implemented and simply stated: if the current training example generates a high error (i.e. greater than a fixed criterion value), it is repeated; otherwise, another one is randomly selected. This approach was motivated by casual observations of behavior in small children; they seem to repeat seemingly arbitrary tasks several times, and then abruptly stop and move to some seemingly arbitrary alternative (Piaget, 1952). For the following discussion, IID and RUB will denote the two selection procedures to be compared.

## 2  METHODOLOGY

RUB can be implemented by adding a condition to the IID statement; in C, this is simply

```
old(IID):  patno = random() % numpats;
new(RUB):   if (paterror<beta) patno = random() % numpats;
```

where *patno* identifies the selected pattern, *numpats* is the number of patterns in the training set, and *paterror* is the sum squared error on a particular pattern. Thus, an example is presented and repeated until it has been learned by the network to some criterion level, the squared error summed across the output units is less than a "boredom" criterion $\beta$; then , another pattern is randomly selected.

The action of RUB in weight space is illustrated in Figure 1, for a two dimensional environment consisting of just two patterns. Corresponding to each pattern, there is an isocline (or equilibrium surface) , defined by the locus of weight vectors that yield the desired response to that pattern (here, **a** or **b**). Since the delta rule drives the weight parallel to the presented pattern, trajectories in weight space are perpendicular to the pattern's isocline. Here, RUB is compared with alternate pattern selection.

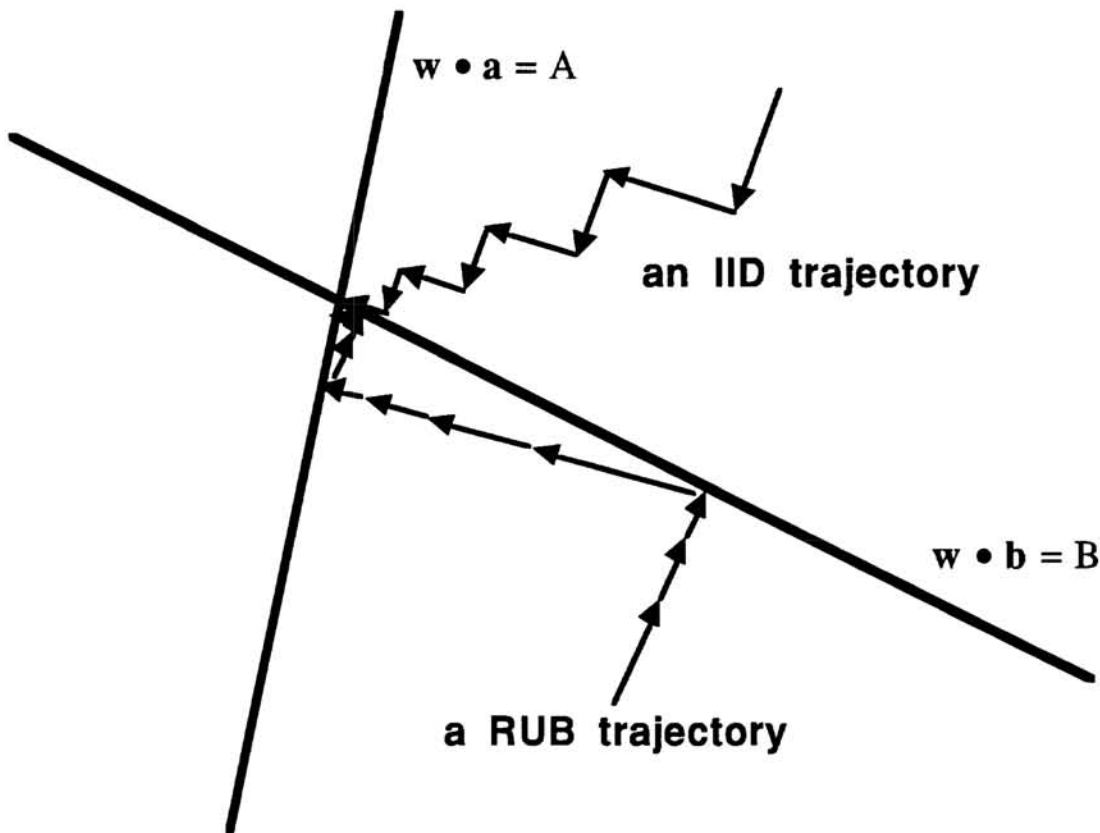

**Figure 1.** *Effect of pattern selection on weight state trajectory.* A linear unit can be trained to give arbitrary responses (A and B) to given stimuli (**a** and **b**). The isoclines (bold lines) are defined to be the set of weights that satisfy each stimulus-response pair. Thus, the intersection is the weight state that satisfies both constraints. The delta rule drives the weights toward the isocline that corresponds to the presented pattern. The RUB procedure repeats a pattern until the state approaches the isocline.

The RUB procedure was tested for a broad range of $\beta$ across several tasks. Two performance measures were used; in both cases, performance was averaged across several (20-100) trials with different initial random weights. For the parity tasks, performance was measured as the fraction of trials for which the squared error summed over the training set reached a sufficiently low value (usually 0.1) within a specified number of training examples. Since the parity task always converged for sufficiently large $\beta$, performance was measured as the number of trials that converged within a prespecified number of iterations required to reduce the total squared error summed across the pattern set to a low value (typically, 0.1). Note that each iteration of weight modification during a set of repeated examples was explicitly counted in the performance measure, so the comparison between IID and RUB is fair. Also, for each task, the learning rate and momentum were fixed (ususally 0.1 and 0.9, respectively).

Consideration of RUB (see the above C implementation, for example) indicates that, for very small values of $\beta$, the first example will be repeated indefinitely, and the task can therefore not be learned. At the other extreme, for $\beta$ greater than or equal to the maximum possible squared error (2.0, in this case), performance should match IID.

# 3  RESULTS

## 3.1.  PARITY

While the expected behavior for RUB on the two and three bit parity tasks (Figure 2) is observed for low and high values of β, there are some surprises in the intermediate range. Rather than proceeding monotonically from zero to its IID value, the performance curve exhibits an "up-down-up" behavior; it reaches a maximum in the range 0.2<β<0.25, then plummets to zero at β=0.25, remains there for an interval, then partially recovers at its final (IID) level. This "dead zone" phenomenon is not as pronounced when the momentum parameter is set to zero (Figure 3).

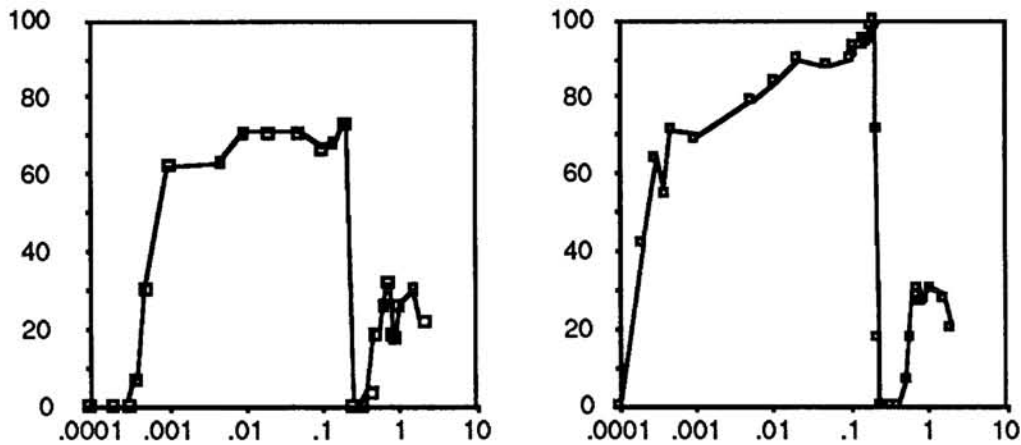

**Figure 2.** *Performance profiles for the parity task.* Each point is the average number of successful simulations out of 100 trials. A log scale is used so that the behavior for very low values of the error criterion is evident. Note the critical falloff at β≈0.25 for both the XOR task (left) and three-bit parity (right).

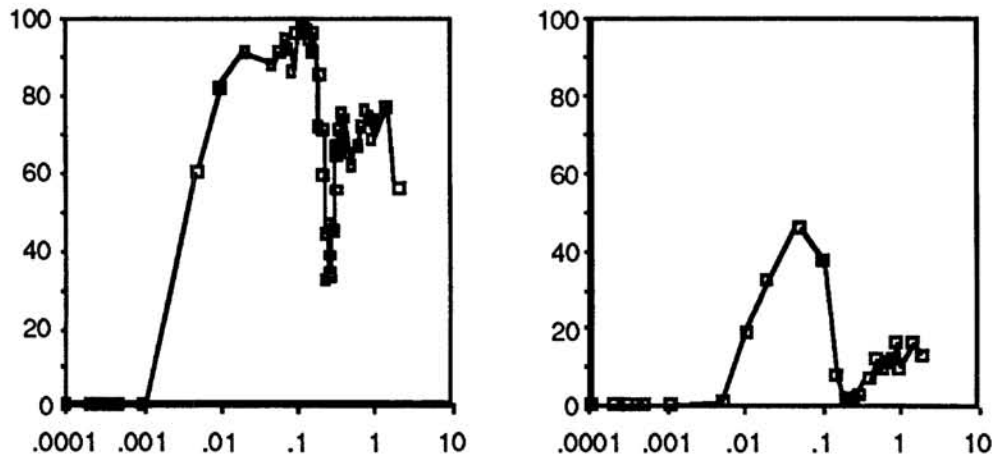

**Figure 3.** *Performance profiles with zero momemtum.* For these two tasks, the up-down-up phenomenon is still evident, but there is no "dead zone". Left: XOR  Right: Three bit parity

## 3.2 ENCODERS

The 4-2-4 encoder shows no significant improvement over the IID for any value of RUB. Here, performance was measured both in terms of success rate and average number of iterations to success. Even though all simulations converge for $\beta > .001$ (i.e., there is no dead zone), the effect of $\beta$ is reflected in another performance measure: average number of iterations to convergence (Figure 4). However, experiments with the 5-2-5 encoder task show an effect. While backprop converges for all values of $\beta$ (except very small values), the performance, as measured by number of pattern presentations, does show a pronounced decrement. The 8-3-8 encoder shows a significant, but less dramatic, effect.

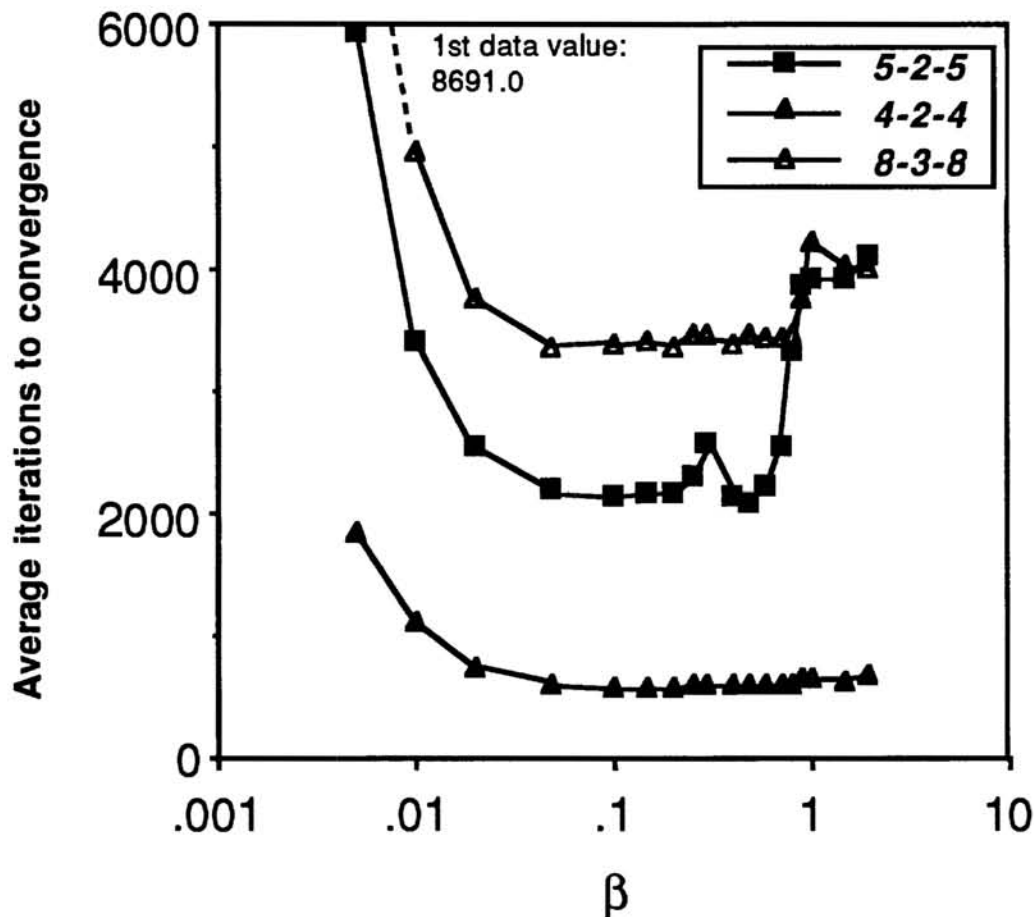

**Figure 4.** *Encoder performance profiles.* See text.

## 3.3 THE MESH

The mesh (Figure 5, left) is a 2-D classification task that can be solved by a strictly lay-ered net with five hidden units. Like the encoder and unlike parity, IID is found to con-verge on 100% of trials; however, there is a critical value of $\beta$ and a well-defined dead zone (Figure 5, right). Note that the curve depicting average number of iterations to con-vergence decreases monotonically, interrupted at the dead zone but continuing its apparent trend for higher values of $\beta$.

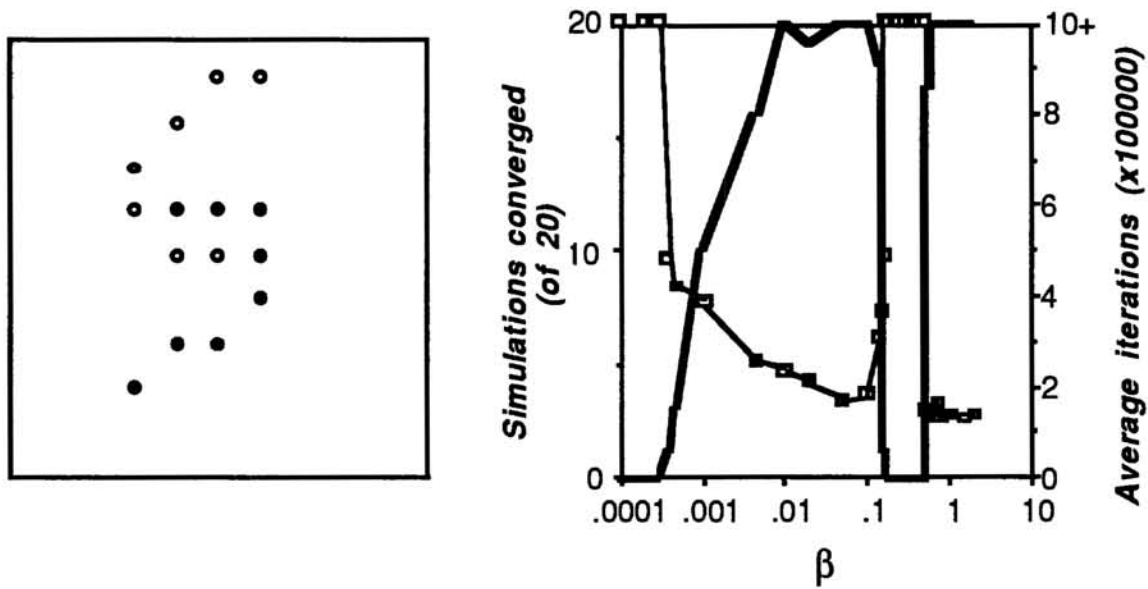

**Figure 5.** *The mesh task.* **Left:** the task. **Right:** Performance profile. Number of simulations that converge is plotted along the bold line (left vertical) axis. Average number of iterations are plotted as squares (right vertical axis).

## 3.4 NONCONVERGENCE

Nonconvergence was examined in detail for three values of $\beta$, corresponding to high performance, poor performance (the dead zone), and IID, for the three bit parity task. The error for each of the eight patterns is plotted over time. For trials that do not converge (Figure 6), the patterns interact differently, depending on the value of $\beta$. At $\beta=0.05$ (a "good" value of $\beta$ for this task), the error traces for the four odd-parity patterns are strongly correlated in an irregular oscillatory mode, as are the four even-parity traces, but the two groups are strongly anticorrelated. In the odd parity group, the error remains low for three of the patterns (001, 010, and 100), but ranges from less than 0.1 to values greater than 0.95 for the fourth (111). Traces for the even parity patterns correspond almost identically; i.e. not only are they correlated, but all four maintain virtually the same value.

At this point, the dead zone phenomenon has only been observed in tasks with a single output unit. This property hints at the following explanation. Note first that each input/output pair in the training set divides the weight space into two halves, characterized by the sign of the linear activation into the output unit; that is, whether the output is above or below 0.5, and hence whether the magnitude of the difference between the actual and desired responses is above or below 0.5. Since $\beta$ is the value of the *squared* error, learning is repeated for $\beta=0.25$ only for examples for which the state is on the wrong *half* of weight space. Just when it is about to cross the category boundary, which would bring the absolute value of the error below .5, RUB switches to another example, and the state is not pushed to the other side of the boundary. This conjecture suggests that for tasks with multiple output units, this effect might be reduced or eliminated, as has been demonstrated in the encoder examples.

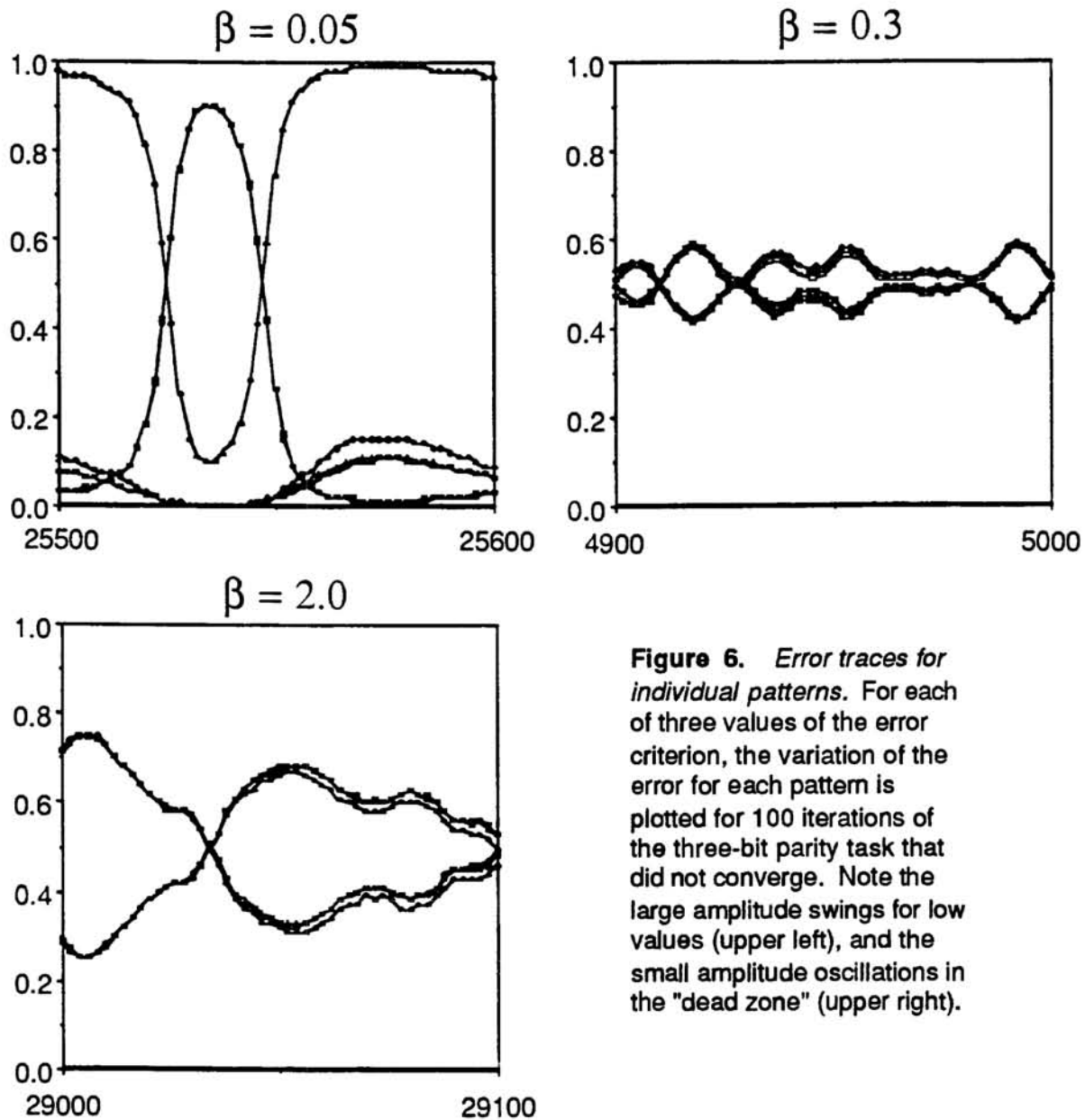

**Figure 6.** *Error traces for individual patterns.* For each of three values of the error criterion, the variation of the error for each pattern is plotted for 100 iterations of the three-bit parity task that did not converge. Note the large amplitude swings for low values (upper left), and the small amplitude oscillations in the "dead zone" (upper right).

## 4   DISCUSSION

*Active learning and boredom.* The sequence of training examples has an undeniable effect on learning, both in the real world and in simulated learning systems. While the RUB procedure influences this sequence such that the learning performance is either positively or negatively affected, it is just a minimal instance of active learning; more elaborate learning systems have explored similar notions of "boredom" (eg., Scott and Markovitch, 1989).

*Nonconvergence.* From Figure 6 it can be seen, for both RUB and IID, that nonconvergence does not correspond to a local minimum in weight space. In situations where the overall error is "stuck" at a non-zero value, the error on the individual patterns continues to change. The weight trajectory is thus "trapped" in a nonoptimal *orbit*, rather than a nonoptimal equilibrium *point*.

## Acknowledgements

This research was supported in part by NSF grant IRI-8910368 and by Siemens Corporate Research, which kindly provided the author with financial support and a stimulating research environment during the summer of 1990. David Cohn and Rik Belew were helpful in bringing relevant work to my attention.

## References

Baum, E. and Lang, K. (1991) Constructing multi-layer neural networks by searching input space rather than weight space. In: *Advances in Neural Information Processing Systems 3*. D. S. Touretsky, ed. Morgan Kaufmann.

Cohn, D., Atlas, L., and Ladner, R. (1990) Training connectionist networks with queries and selective sampling. In: *Advances in Neural Information Processing Systems 2*. D. S. Touretsky, ed. Morgan Kaufmann.

Mozer, M. and Bachrach, J. (1990) Discovering the structure of a reactive environment by exploration. In: *Advances in Neural Information Processing Systems 2*. D. S. Touretsky, ed. Morgan Kaufmann.

Parker, D. (1985) Learning logic. TR-47. MIT Center for Computational Economics and Statistics. Cambridge MA.

Piaget, J. (1952) *The Origins of Intelligence in Children*. Norton.

Rumelhart D., Hinton G., and Williams R. (1986) Learning representations by back-propagating errors. *Nature* 323:533-536.

Scott, P. D. and Markovitch, S. (1989) Uncertainty based selection of learning experiences. *Sixth International Workshop on Machine Learning*. pp.358-361

Werbos, P. (1974) Beyond regression: new tools for prediction and analysis in the behavioral sciences. Unpublished doctoral dissertation, Harvard University.